# Morphogenesis of the Lateral Geniculate Nucleus: How Singularities Affect Global Structure

**Svilen Tzonev**
Beckman Institute
University of Illinois
Urbana, IL 61801
svilen@ks.uiuc.edu

**Klaus Schulten**
Beckman Institute
University of Illinois
Urbana, IL 61801
kschulte@ks.uiuc.edu

**Joseph G. Malpeli**
Psychology Department
University of Illinois
Champaign, IL 61820
jmalpeli@uiuc.edu

## Abstract

The macaque lateral geniculate nucleus (LGN) exhibits an intricate lamination pattern, which changes midway through the nucleus at a point coincident with small gaps due to the blind spot in the retina. We present a three-dimensional model of morphogenesis in which local cell interactions cause a wave of development of neuronal receptive fields to propagate through the nucleus and establish two distinct lamination patterns. We examine the interactions between the wave and the localized singularities due to the gaps, and find that the gaps induce the change in lamination pattern. We explore critical factors which determine general LGN organization.

## 1 INTRODUCTION

Each side of the mammalian brain contains a structure called the lateral geniculate nucleus (LGN), which receives visual input from both eyes and sends projections to

the primary visual cortex. In primates the LGN consists of several distinct layers of neurons separated by intervening layers of axons and dendrites. Each layer of neurons maps the opposite visual hemifield in a topographic fashion. The cells comprising these layers differ in terms of their type (magnocellular and parvocellular), their input (from ipsilateral (same side) and contralateral (opposite side) eyes), and their receptive field organization (ON and OFF center polarity). Cells in one layer receive input from one eye only (Kaas et al., 1972), and in most parts of the nucleus have the same functional properties (Schiller & Malpeli, 1978). The maps are in register, i.e., representations of a point in the visual field are found in all layers, and lie in a narrow column roughly perpendicular to the layers (Figure 1). A prominent

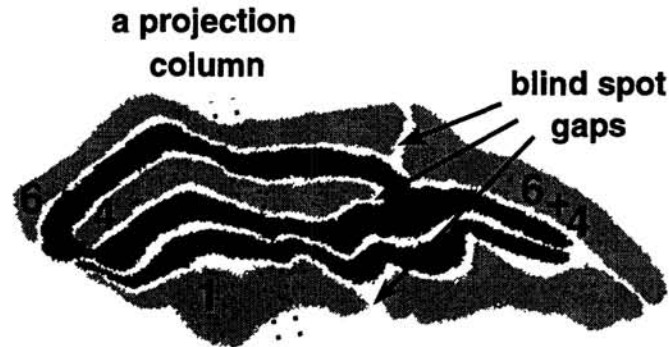

Figure 1: A slice along the plane of symmetry of the macaque LGN. Layers are numbered ventral to dorsal. Posterior is to the left, where foveal (central) parts of the retinas are mapped; peripheral visual fields are mapped anteriorly (right). Cells in different layers have different morphology and functional properties: 6-P/C/ON; 5-P/I/ON; 4-P/C/OFF; 3-P/I/OFF; 2-M/I/ON&OFF; 1-M/C/ON&OFF, where P is parvocellular, M is magnocellular, C is contralateral, I is ipsilateral, ON and OFF refer to polarities of the receptive-field centers. The gaps in layers 6, 4, and 1 are images of the blind spot in the contralateral eye. Cells in columns perpendicular to the layers receive input from the same point in the visual field.

feature in this laminar organization is the presence of cell-free gaps in some layers. These gaps are representations of the blind spot (the hole in the retina where the optic nerve exits) of the opposite retina. In the LGN of the rhesus macaque monkey (*Macaca mulatta*) the pattern of laminar organization drastically changes at the position of the gaps — foveal to the gaps there are six distinct layers, peripheral to the gaps there are four layers. The layers are extended two-dimensional structures whereas the gaps are essentially localized. However, the laminar transition occurs in a surface that extends far beyond the gaps, cutting completely across the main axis of the LGN (Malpeli & Baker., 1975).

We propose a developmental model of LGN laminar morphogenesis. In particular, we investigate the role of the blind-spot gaps in the laminar pattern transition, and their extended influence over the global organization of the nucleus. In this model a wave of development caused by local cell interactions sweeps through the system (Figure 2). Strict enforcement of retinotopy maintains and propagates an initially localized foveal pattern. At the position of the gaps, the system is in a metastable

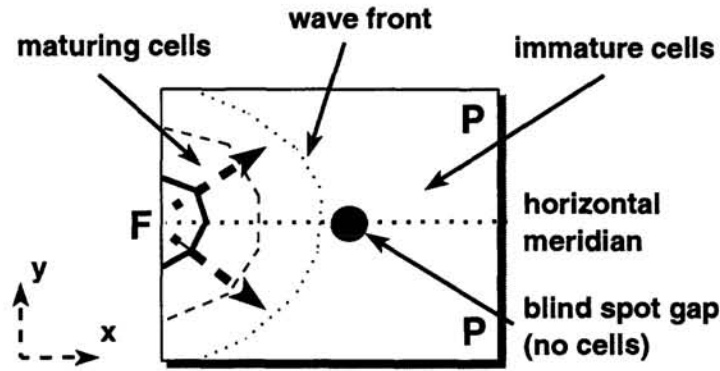

Figure 2: Top view of a single layer. As a wave of development sweeps through the LGN the foveal part matures first and the more peripheral parts develop later. The shape of the developmental wave front is shown schematically by lines of "equal development".

state, and the perturbation in retinotopy caused by the gaps is sufficient to change the state of the system to its preferred four-layered pattern. We study the critical factors in this model, and make some predictions about LGN morphogenesis.

## 2   MODEL OF LGN MORPHOGENESIS

We will consider only the upper four (parvocellular) layers since the laminar transition does not involve the other two layers. This transition results simply from a reordering of the four parvocellular strata (Figure 1). Foveal to the gaps, the strata form four morphologically distinct layers (6, 5, 4 and 3) because adjacent strata receive inputs from opposite eyes, which "repel" one another. Peripheral to the gaps, the reordering of strata reduces the number of parvocellular eye alternations to one, resulting in two parvocellular layers (6+4 and 5+3).

### 2.1   GEOMETRY AND VARIABLES

LGN cells $c_i$ are labeled by indices $i = 1, 2, \ldots, N$. The cells have fixed, quazi-random and uniformly distributed locations $r_i \in V \subset \Re^3$, where $V = \{ (x, y, z) \,|\, 0 < x < S_x, 0 < y < S_y, 0 < z < S_z \}$, and belong to one projection column $C_{ab}$, $a = 1, 2, \ldots, A$ and $b = 1, 2, \ldots, B$, (Figure 3). Functional properties of the neurons change in time (denoted by $\tau$), and are described by eye specificity and receptive-field polarity, $e_i(\tau)$, and $p_i(\tau)$, respectively: $e_i(\tau), p_i(\tau) \in [-1, 1] \subset \Re$, $i = 1, 2, \ldots, N$, $\tau = 0, 1, \ldots, T_{max}$.

The values of eye specificity and polarity represent the proportions of synapses from competing types of retinal ganglion cells (there are four type of ganglion cells — from different eyes and with ON or OFF polarity). $e_i = -1$ ($e_i = 1$) denotes that the $i$-th cell is receiving input solely from the opposite (same side) retina. Similarly, $p_i = -1$ ($p_i = 1$) denotes that the cell input is pure ON (OFF) center. Intermediate values of $e_i$ and $p_i$ imply that the cell does not have pure properties (it receives

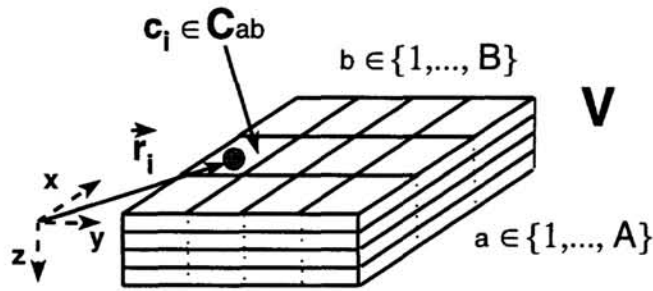

Figure 3: Geometry of the model. LGN cells $c_i$ $(i = 1, 2, \ldots, N)$ have fixed random, and uniformly-distributed locations $r_i$ within a volume $V \subset \Re^3$, and belong to one projection column $C_{ab}$.

input from retinal ganglion cells of both eyes and with different polarities). Initially, at $\tau = 0$, all LGN cells are characterized by $e_i$, $p_i = 0$. This corresponds to two possibilities: no retinal ganglion cells synapse on any LGN cell, or proportions of synapses from different ganglion cells on all LGN neurons are equal, i.e., neurons possess completely undetermined functionality because of competing inputs of equal strength. As the neurons mature and acquire functional properties, their eye specificity and polarity reach their asymptotic values, $\pm 1$.

Even when cells are not completely mature, we will refer to them as being of four different types, depending on the signs of their functional properties. Following accepted anatomical notation, we will label them as 6, 5, 4, and 3. We denote eye specificity of cell types 6 and 4 as negative, and cell types 5 and 3 as positive. Polarity of cell types 6 and 5 is negative, while polarity of types 4 and 3 is positive.

Cell functional properties are subject to the dynamics described in the following section. The process of LGN development starts from its foveal part, since in the retina it is the fovea that matures first. As more peripheral parts of the retina mature, their ganglion cells start to compete to establish permanent synapses on LGN cells. In this sorting process, each LGN cell gradually emerges with permanent synapses that connect only to several neighboring ganglions of the same type. A wave of gradual development of functionality sweeps through the nucleus. The driving force for this maturation process is described by localized cell interactions modulated by external influences. The particular pattern of the foveal lamination is shaped by external forces, and later serves as a starting point for a "propagation of sameness" of cell properties. Such a sameness propagation produces clustering of similar cells and formation of layers. It should be stressed that cells do not move, only their characteristics change.

## 2.2   DYNAMICS

The variables describing cell functional properties are subject to the following dynamics

$$
\begin{aligned}
\hat{e}_i\,(\tau + 1) &= e_i\,(\tau) + \Delta e_i\,(\tau) + \eta_e \\
\hat{p}_i\,(\tau + 1) &= p_i\,(\tau) + \Delta p_i\,(\tau) + \eta_p, \quad i = 1, 2, \ldots, N.
\end{aligned} \tag{1}
$$

In Eq. (1), there are two contributions to the change of the intermediate variables $\hat{e}_i(\tau)$ and $\hat{p}_i(\tau)$. The first is deterministic, given by

$$\Delta e_i(\tau) = \alpha(r_i)\left[\left(\sum_{j=1}^{N} e_j(\tau) f(|r_i - r_j|)\right) + E_{ext}(r_i)\right](1 - e_i^2(\tau))\,\beta_{ab}^t$$

$$\Delta p_i(\tau) = \alpha(r_i)\left[\left(\sum_{j=1}^{N} p_j(\tau) f(|r_i - r_j|)\right) + P_{ext}(r_i)\right](1 - p_i^2(\tau))\,\beta_{ab}^t. \quad (2)$$

The second is a stochastic contribution corresponding to fluctuations in the growth of the synapses between retinal ganglion cells and LGN neurons. This noise in synaptic growth plays both a driving and a stabilizing role to be explained below. We explain the meaning of the variables in Eq. (2) only for the eye specificity variable $e_i$. The corresponding parameters for polarity $p_i$ have similar interpretations.

The parameter $\alpha(r_i)$ is the rate of cell development. This rate is the same for eye specificity and polarity. It depends on the position $r_i$ of the cell in order to allow for spatially non-uniform development. The functional form of $\alpha(r_i)$ is given in the Appendix.

The term $E_{int}(r_i) = \sum_{j=1}^{N} e_j f(|r_i - r_j|)$ is effectively a cell force field. This field influences the development of nearby cells and promotes clustering of same type of cells. It depends on the maturity of the generating cells and on the distance between cells through the interaction function $f(\delta)$. We chose for $f(\delta)$ a Gaussian form, i.e., $f(\delta) = \exp(-\delta^2/\sigma^2)$, with characteristic interaction distance $\sigma$.

The external influences on cell development are incorporated in the term for the external field $E_{ext}(r_i)$. This external field plays two roles: it launches a particular laminar configuration of the system (in the foveal part of the LGN), and determines its peripheral pattern. It has, thus, two contributions $E_{ext}(r_i) = E_{ext}^f(r_i) + E_{ext}^p(r_i)$. The exact forms of $E_{ext}^f(r_i)$ and $E_{ext}^p(r_i)$ are provided in the Appendix.

The nonlinear term $(1 - e_i^2)$ in Eq. (2) ensures that $\pm 1$ are the only stable fixed points of the dynamics. The neuronal properties gradually converge to either of these fixed points capturing the maturation process. This term also stabilizes the dynamic variables and prevents them from diverging.

The last term $\beta_{ab}^t(\tau)$ reflects the strict columnar organization of the maps. At each step of the development the proportion of all four types of LGN cells is calculated within a single column $C_{ab}$, and $\beta_{ab}^t(\tau)$ for different types $t$ is adjusted such that all types are equally represented. Without this term, the cell organization degenerates to a non-laminar pattern (the system tries to minimize the surfaces between cell clusters of different type). The exact form of $\beta_{ab}^t(\tau)$ is given in the Appendix.

At each stage of LGN development, cells receive input from retinal ganglion cells of particular types. This means that eye specificity and polarity of LGN cells are not independent variables. In fact, they are tightly coupled in the sense that $|e_i(\tau)| = |p_i(\tau)|$ should hold for all cells at all times. This gives rise to coupled dynamics described by

$$
\begin{aligned}
m_i(\tau+1) &= \min\left(|\hat{e}_i(\tau+1)|, |\hat{p}_i(\tau+1)|\right) \\
e_i(\tau+1) &= m_i(\tau+1)\,\mathrm{sgn}\left(\hat{e}_i(\tau+1)\right) \\
p_i(\tau+1) &= m_i(\tau+1)\,\mathrm{sgn}\left(\hat{p}_i(\tau+1)\right), \; i=1,2,\ldots,N.
\end{aligned}
\tag{3}
$$

The blind spot gaps are modeled by not allowing cells in certain columns to acquire types of functionality for which retinal projections do not exist, e.g., from the blind spot of the opposite eye. Accordingly, $e_i$ is not allowed to become negative. Thus, some cells never reach a pure state $e_i, p_i = \pm 1$. It is assumed that in reality such cells die out. Of all quantities and parameters, only variables describing the neuronal receptive fields ($e_i$ and $p_i$) are time-dependent.

## 3  RESULTS

We simulated the dynamics described by Eqs. (1, 2, 3), typically for $100,000$ time steps. Depending on the rate of cell development, mature states were reached in about $10,000$ steps. The maximum value of $\alpha$ was 0.0001. We used an interaction function with $\sigma = 1$.

First, we considered a two-dimensional LGN, $V = \{(x,z)|0 < x < S_x, 0 < z < S_z\}$ with $S_x = 10$ and $S_z = 6$. There were ten projection columns (with equal size) along the $x$ axis. An initial pattern was started in the foveal part by the external field. The size of the gaps $g$ measured in terms of the interaction distance $\sigma$ was crucial for pattern development. When the developmental wave reached the gaps, layer 6 could "jump" its gap and continued to spread peripherally if the gap was sufficiently narrow ($g/\sigma < 1.5$). If its gap was not too narrow ($g/\sigma > 0.5$), layer 4 completely stopped (since cells in the gaps were not allowed to acquire negative eye specificity) and so layers 5 and 3 were able to merge. Cells of type 4 reappeared after the gaps (Figure 4, right side, shows behavior similar to the two-dimensional model) because of the required equal representation of all cell types in the projection columns, and because of noise in cell development. Energetically, the most favorable position of cell type 4 would be on top of type 6, which is inconsistent with experimental observations. Therefore, one must assume the existence of an external field in the peripheral part that will drive the system away from its otherwise preferred state. If the gaps were too large ($g/\sigma > 1.5$), cells of type 6 and 4 reappeared after the gaps in a more or less random vertical position and caused transitions of irregular nature. On the other hand, if the gaps were too narrow ($g/\sigma < 0.5$), both layers 6 and 4 could continue to grow past their gaps, and no transition between laminar patterns occurred at all. When $g/\sigma$ was close to the above limits, the pattern after the gaps differed from trial to trial. For the two-dimensional system, a realistic peripheral pattern always occurred for $0.7 < g/\sigma < 1.2$.

We simulated a three-dimensional system with size $S_x = 10$, $S_y = 10$, and $S_z = 6$, and projection columns ordered in a 10 by 10 grid. The topology of the system is different in two and three dimensions: in two dimensions the gaps interrupt the layers completely and, thus, induce perturbations which cannot be by-passed. In three dimensions the gaps are just holes in a plane and generate localized perturbations: the layers can, in principle, grow around the gaps maintaining the initial laminar pattern. Nevertheless, in the three-dimensional case, an extended transi-

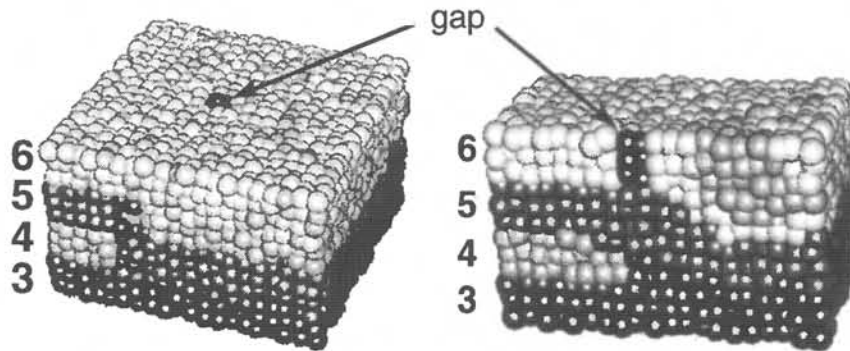

Figure 4: **Left:** Mature state of the macaque LGN — result of the three-dimensional model with 4,800 cells. Spheres with different shades represent cells with different properties. Gaps in strata 6 and 4 (this gap is not visible) are coded by the darkest color, and coincide with the transition surface between 4- and 2-layered patterns. **Right:** A cut of the three-dimensional structure along its plane of symmetry. A two-dimensional system exhibits similar organization. Compare with upper layers in Figure 1. Spatial segregation between layers is not modeled explicitly.

tion was triggered by the gaps. The transition surface, which passed through the gaps and was oriented roughly perpendicularly to the $x$ axis, cut completely across the nucleus (Figure 4).

Several factors were critical for the general behavior of the system. As in two dimensions, the size of the gaps must be within certain limits: typically $0.5 < g/\sigma < 1.0$. These limits depend on the curvature of the wavefront. The gaps must lie in a certain "inducing" interval along the $x$ axis. If they were too close to the origin, the foveal pattern was still more stable, so no transition could be induced there. However, a spontaneous transition might occur downstream. If the gaps were too far from the origin, a spontaneous transition might occur before them. The occurrence and location of a spontaneous transition , (therefore, the limits of the "inducing" interval) depended on the external-field parameters. A realistic transition was observed only when the front of the developmental wave had sufficient curvature when it reached the gaps. Underlying anatomical reasons for a sufficiently curved front along the main axis could be the curvature of the nucleus, differences in layer thickness, or differences in ganglion-cell densities in the retinas.

Propagation of the developmental wave away from the gaps was quite stable. Before and after the gaps, the wave simply propagated the already established patterns. In a system without gaps, transitions of variable shape and location occurred when the peripheral contribution to the external fields was sufficiently large; a weaker contribution allowed the foveal pattern to propagate through the entire nucleus.

## 4 SUMMARY

We present a model that successfully captures the most important features of macaque LGN morphogenesis. It produces realistic laminar patterns and supports

the hypothesis (Lee & Malpeli, 1994) that the blind spot gaps trigger the transition between patterns. It predicts that critical factors in LGN development are the size and location of the gaps, cell interaction distances, and shape of the front of the developmental wave. The model may be general enough to incorporate the LGN organizations of other primates. Small singularities, similar to the blind spot gaps, may have an extended influence on global organization of other biological systems.

## Acknowledgements

This work has been supported by a Beckman Institute Research Assistantship, and by grants PHS 2P41 RR05969 and NIH EY02695.

## References

J.H. Kaas, R.W. Guillery & J.M. Allman. (1972) Some principles of organization in the dorsal lateral geniculate nucleus, *Brain Behav. Evol.*, **6**: 253-299.

D.Lee & J.G.Malpeli. (1994) Global Form and Singularity: Modeling the Blind Spot's Role in Lateral Geniculate Morphogenesis, *Science*, **263**: 1292-1294.

J.G. Malpeli & F.H. Baker. (1975) The representation of the visual field in the lateral geniculate nucleus of Macaca mulatta, *J. Comp. Neurol.*, **161**: 569-594.

P.H. Schiller & J.G. Malpeli. (1978) Functional specificity of LGN of rhesus monkey, *J. Neurophysiol.*, **41**: 788-797.

## APPENDIX

The form of $\alpha(x, y, z)$ (with $\alpha_0 = 0.0001$) was chosen as

$$\alpha(x, y, z) = \alpha_0 \left(0.1 + \exp\left(-(y - S_y/2)^2\right)\right). \tag{4}$$

Foveal external fields of the following form were used:

$$
\begin{aligned}
E_{ext}^f(x,y,z) &= 10\left[\theta(z-d) - 2\theta(z-2d) + 2\theta(z-3d) - \theta(d-z)\right]\exp(-x) \\
P_{ext}^f(x,y,z) &= 10\left[2\theta(z-2d) - 1\right]\exp(-x),
\end{aligned} \tag{5}
$$

where $d = S_z/4$ is the layers' thickness and the "theta" function is defined as $\theta(x) = 1, x > 0$ and $\theta(x) = 0, x < 0$. Peripheral external fields (in fact they are present everywhere but determine the pattern in the peripheral part only) were chosen as

$$
\begin{aligned}
E_{ext}^p(x,y,z) &= 5\left[2\theta(z-2d) - 1\right] \\
P_{ext}^p(x,y,z) &= 5\left[\theta(z-d) - 2\theta(z-2d) + 2\theta(z-3d) - \theta(d-z)\right]. \tag{6}
\end{aligned}
$$

$\beta_{ab}^t(\tau)$ was calculated in the following way: at any given time $\tau$, within the column $C_{ab}$, we counted the number $N_{ab}^t(\tau)$ of cells, that could be classified as one of the four types $t = 3, 4, 5, 6$. Cells with $e_i(\tau)$ or $p_i(\tau)$ exactly zero were not counted. The total number of classified cells is then $N_{ab}(\tau) = \sum_{t=3}^{6} N_{ab}^t(\tau)$. If there were no classified cells ($N_{ab}(\tau) = 0$), then $\beta_{ab}^t(\tau)$ was set to one for all $t$. Otherwise the ratio of different types was calculated: $n_{ab}^t = N_{ab}^t(\tau)/N_{ab}(\tau)$. In this way we calculated

$$\beta_{ab}^t(\tau) = 4 - 12\, n_{ab}, \quad t = 3, 4, 5, 6. \tag{7}$$

If $\beta_{ab}^t(\tau)$ was negative it was replaced by zero.